# An Integrated Architecture of Adaptive Neural Network Control for Dynamic Systems

Liu Ke[1,2]                    Robert L. Tokar[2]                    Brian D.McVey[2]

[1]Center for Nonlinear Studies, [2]Applied Theoretical Physics Division
Los Alamos National Laboratory,    Los Alamos, NM, 87545

## Abstract

In this study, an integrated neural network control architecture for nonlinear dynamic systems is presented. Most of the recent emphasis in the neural network control field has no error feedback as the control input, which rises the lack of adaptation problem. The integrated architecture in this paper combines feed forward control and error feedback adaptive control using neural networks. The paper reveals the different internal functionality of these two kinds of neural network controllers for certain input styles, e.g., state feedback and error feedback. With error feedback, neural network controllers learn the slopes or the gains with respect to the error feedback, producing an error driven adaptive control systems. The results demonstrate that the two kinds of control scheme can be combined to realize their individual advantages. Testing with disturbances added to the plant shows good tracking and adaptation with the integrated neural control architecture.

## 1 INTRODUCTION

Neural networks are used for control systems because of their capability to approximate nonlinear system dynamics. Most neural network control architectures originate from work presented by Narendra[1], Psaltis[2] and Lightbody[3]. In these architectures, an identification neural network is trained to function as a model for the plant. Based on the neural network identification model, a neural network controller is trained by backpropagating the error through the identification network. After training, the identification network is replaced by the real plant. As is illustrated in Figure 1, the controller receives external inputs as well as plant state feedback inputs. Training procedures are employed such that the networks approximate feed forward control surfaces that are functions of external inputs and state feedbacks of the plant (or the identification network during training).

It is worth noting that in this architecture, the error between the plant output and the desired output of the reference model is not fed back to the controller, after the training phase. In other words, this error information is ignored when the neural network applies its control. It is well known in control theory that the error feedback plays a significant role in adaptation. Therefore, when model uncertainty or noise/disturbances are present, a feed forward neural network controller with only state feedback will not adaptively update the control signal. On line training for the neural controller has been proposed to obtain adaptive ability[1][3]. However, the stability for the on line training of the neural network controller is unresolved[1][4].

In this study, an additional nonlinear recurrent network is combined with the feed forward neural network controller to form an adaptive controller. This added neural network uses feedback error between the reference model output and the plant output as an input. In addition, the system's external

inputs and the plant states are also input to the feedback network. This architecture is used in the control community, but not with neural network components. The approach differs from a conventional error feedback controller, such as a gain scheduled PID controller, in that the neural network error feedback controller implements a continuous nonlinear gain scheduled hypersurface, and after training, adaptive model reference control for nonlinear dynamic systems is achieved without further parameter computation. The approach is tested on well-known nonlinear control problems in the neural network literature, and good results are obtained.

## 2 NEURAL NETWORK CONTROL

In this section, several different neural network control architectures are presented. In these structures, identification neural networks, viewed as accurate models for real plants, are used.

### 2.1 NEURAL NETWORK FEED FORWARD CONTROL

The neural network controllers are trained by backpropagation of errors through a well trained neural identification network. In this architecture, the state variable $y(t)$ of the system is sent back to the neural network, and the external input $x(t)$ also is input to the network. With these inputs, the neural network establishes a feed forward mapping from the external input $x(t)$ to the control signal $u(t)$. This control mapping is expressed as a function of the external input $x(t)$ and the plant state $y(t)$:

$$u(t)=f(x(t), y(t)) \tag{1}$$

where $x(t)=[x(t), x(t-1), ...]^T$, and $y(t)=[y(t), y(t-1), ...]^T$.

This neural network control architecture is denoted in this study as feed forward neural control even though it includes state feedback . Neural control with error feedback is denoted as feedback neural control.

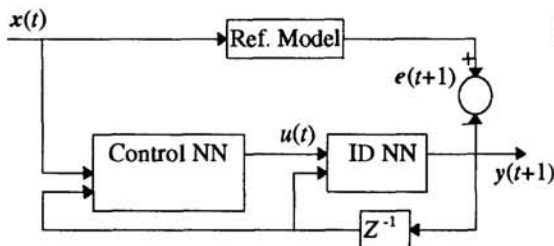
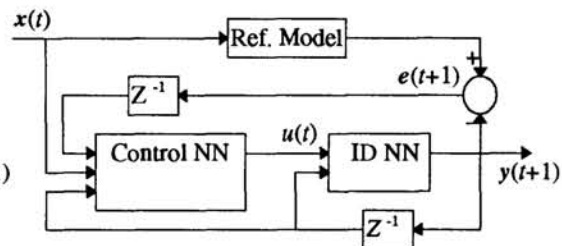

Figure 1 Neural Network Control Architecture.
ID NN represents the identification network.
Ref. Model means reference model, and NN
means neural network.

Figure 2 Neural Network Feedback Control
Architecture

During the training phases, based on the assumption that the neural identification network provides a model for the plant, the gradient information needed for error backpropagation is obtained by calculating the Jacobian of the identification network. The following equation describes this process for the control architecture shown in Figure 1. If the cost function is defined as $E$, then the gradient of the cost function with respect to weight $w$ of the neural controller is

$$\frac{\partial E}{\partial w} = \frac{\partial E}{\partial u}\frac{\partial u}{\partial w} + \left(\frac{\partial E}{\partial u}\frac{\partial u}{\partial y_{t-1}} + \frac{\partial E}{\partial y_{t-1}}\right)\frac{\partial y_{t-1}}{\partial w} \qquad (2)$$

where $u$ is the control signal and $y_{t-1}$ is the plant feedback state.

After the training stage, the neural network supplies a control law. Because neural networks have the ability to approximate any arbitrary nonlinear functions[5], a feed forward neural network can build a nonlinear controller, which is crucial to the use of the neural network in control engineering. Also, since all the parameters of the neural network identification model and the neural network controller are obtained from learning through samples, mathematically untraceable features of the plant can be extracted from the samples and imbedded into the control system.

However, because the feed forward controller has no error feedback, the controller can not adapt to the disturbances occurring in the plant or the reference model. This problem is of substantial importance in the context of adaptive control. In the next subsection, error feedback between the reference models and the plant outputs is introduced into neural network controllers for adaptation.

## 2.2 NEURAL ADAPTIVE CONTROL WITH ERROR FEEDBACK

It is known that feedback errors from the system are important for adaptation. Due to the flexibility of the neural network architecture, the error between the reference model and the plant can be sent back to the controller as an extra input. In such an architecture, neural networks become nonlinear gain scheduled controllers with smooth continuous gains. Figure 2 shows the architecture for the feedback neural control.

With this architecture, the neural network control surface is not the fixed mapping from the $x(t)$ to $u(t)$ for each state $y(t)$, but instead it learns the *slope* or the *gain* referring to the feedback error $e(t)$ for control. This gain is a continuous nonlinear function of the external input $x(t)$ and the state feedback $y(t)$. Figure 3 shows the recurrent network architecture of the feedback neural controller. The output node needs to be recurrent because the output without the recurrent link from the neural controller is only a correction to the old control signal, and the new control signal should be the combination of old control signal and the correction. The other nodes of the network can be feed forward or recurrent. If we denote the weight for the output node's recurrent link as $w_b$, then the output from the recurrent link is $w_b u(t-1)$. The following equation describes the feedback network.

$$u(t) = w_b u(t-1) + f(x(t), y(t), e(t)) \qquad (3)$$

where $f(.)$ is a nonlinear function established by the network for which the recurrent link output is not included and $e(t) = [e(t), e(t-1), ...]^T$.

To compare the control gain expression with conventional control theory, consider the Taylor series expansion of the network forward mapping $f(.)$, equation (3) becomes

$$u(t) = w_b u(t-1) + f'(x(t), y(t)) e(t) + f''(x(t), y(t)) e^2(t) + ... \qquad (4)$$

where $f'(x(t), y(t)) = [\partial f(x(t), y(t), e(t))/\partial e(t), \partial f(x(t), y(t), e(t))/\partial e(t-1), ...]$. If high order terms are ignored and $g(.)$ represents $f'(.)$, we get

$$u(t) = w_b u(t-1) + g(x(t), y(t)) e(t) \qquad (5)$$

which is a gain scheduled controller and the gain is the function of external input $x(t)$ and the plant state $y(t)$. It is clear that when $w_b=1.0$, $g(.)$ is a constant vector and $e(t)=[e(t), e(t-1), e(t-2)]^T$, the feedback neural network controller degenerates to a discrete PID controller. Because the neural network can approximate arbitrary nonlinear functions through learning, the neural network feedback controller can generate a nonlinear continuous gain hypersurface.

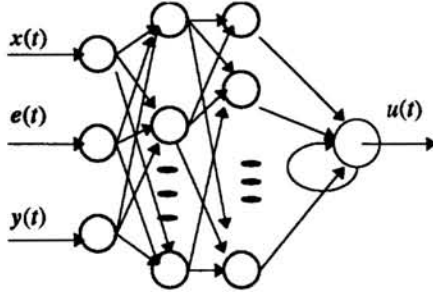
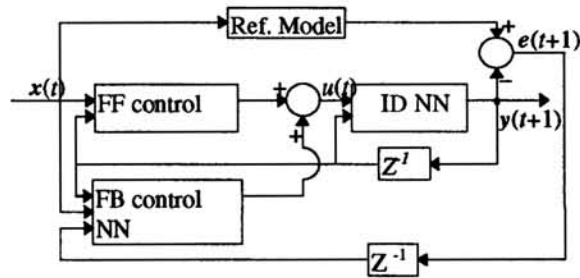

Figure 3 Feedback Neural Network Controller          Figure 4 Integrated NN Control Architeture.

In the training process, error backpropagating through the identification network is used. The process is similar to the training of a feed forward neural controller, but the resulting control surface is completely different due to the different inputs. After training, the neural network is able to provide a nonlinear control law, that is, the desired model following response can be obtained with fixed controller parameters for nonlinear dynamic systems. Traditionally, the control of the nonlinear plant is derived from continuous computing of the controller gains.

This feedback controller is error driven. As long as an error exists, the control signal is updated according to the error and the *gain*. This kind of neural controller is an adaptive controller in principle.

## 2.3 INTEGRATED NEURAL NETWORK CONTROLLER

The characteristics of feed forward and error feedback neural control networks are described in the previous subsections. In this section, the two controllers are combined. Figure 4 shows the architecture.

In this architecture, we include both feed forward and feedback neural network controllers. The control signal is the combination from these two networks' outputs. In the training stage, it is our experience that the feed forward network should be trained first. The feedback network is not included while training the feed forward network. After training the feed forward controller, the error feedback network is trained with the feed forward network, but the feed forward networks' weights are unchanged. Backpropagating the error through the identification network is applied for the training of both networks.

When training the feedback control network, the feed forward calculation is

$$u(t) = u_{ff}(t)+u_{fb}(t), \tag{6}$$

$$y(t+1) = P(x(t), y(t), u(t)), \tag{7}$$

where $u_{ff}(t)$ is the output from the feed forward controller network and $u_{fb}(t)$ is the output from the feedback controller network, $P(.)$ is the identification mapping.

## 3 CONTROL ON EXAMPLE PROBLEMS

In this section, the control architecture described above is applied to a well-known problem from the literature[1]. The plants and the reference model of the sample problems are described by difference equations

$$\text{plant:} \qquad y(t+1) = \frac{y(t)}{1.0 + y^2(t)} + (u(t) - 1.0)u(t)(u(t) + 1.0) \qquad (11)$$

$$\text{reference model:} \qquad y(t+1) = 0.6y(t) + u(t) \qquad (12)$$

This is a nonlinear time varying dynamic system with no analytical inverse.

### 3.1 FEED FORWARD CONTROL

A feed forward neural network is trained to control the system to follow the reference model. The plant state $y(t)$ and external input $x(t)$ are fed to the controller. During the training, the $x(t)$ is randomly generated. After training, the controller generates a control signal $u(t)$ such that the plant can follow the reference model output. Figure 5 shows the testing result of the reference model output and the controlled plant output. The input function is $x(t) = \sin(2\pi t/25) + \sin(2\pi t/10)$. The controller network architecture is (2, 20, 1).

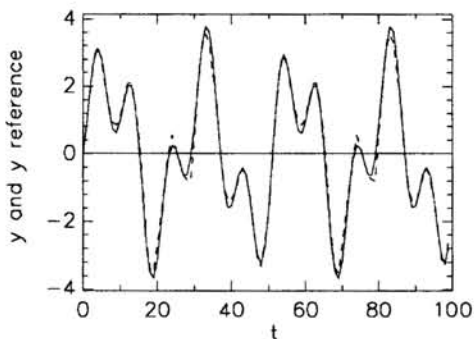
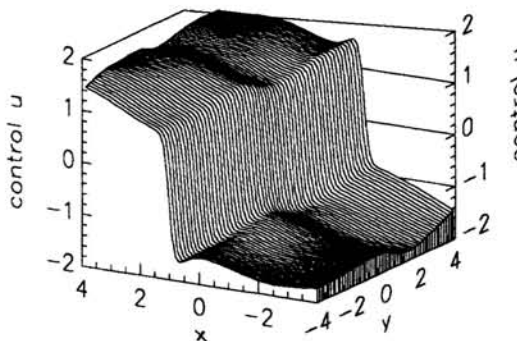

Figure 5 Tracking Result From the Feed Forward NN. Output of reference (solid line) and plant (dash line).

Figure 6 Feed Forward Control Surface

The output surface of the controller network is shown in Figure 6. By examining the controller output surface, we can see that the neural network builds a feed forward mapping from $x(t)$ to $u(t)$. This feed forward mapping is also a function of the plant state $y(t)$. Under each state, the neural network controller accepts input $x(t)$ to produce control signal $u(t)$ such that the plant follows the reference model reasonably well. In Figure 6, the $x$ axis is the external input $x(t)$ and the $y$ axis is the plant feedback output $y(t)$. The $z$ axis represents the control surface.

The feed forward controller lacks the ability to adapt to plant uncertainty, noise or changes in the reference model. As an example, we apply this feed forward controller to the disturbed plant with a bias 0.5 added to the original plant. The tracking result is shown in Figure 7. With this slight bias, the plant does not follow the reference model. Clearly, the feed forward controller has no adaptive ability to this model bias.

### 3.2 FEEDBACK CONTROL

First, we compare the neural network feedback controller with fixed gain PID controllers. For many nonlinear systems, the fixed gain PID controllers will give poor tracking and continuous adaptation of the controller parameters is needed. The neural network approach offers an alternative control approach for nonlinear systems. Through the training, control gains, imbedded in the neural network, are established as a continuous function of system external inputs $x(t)$ and plant states $y(t)$.

The sample problem in the above section is now employed to describe how the neural network creates a nonlinear control gain surface with error feedback and additional inputs. First, we show one simple case of neural adaptive feedback controller. This controller can only adapt to the system nonlinearity with a fixed linear input pattern. The reason to show this simple adaptation case first is that its control gain surface can be illustrated graphically.

Figure 8 illustrates, for the system in equations (11) and (12) that a fixed gain PI controller fails to track the reference model, for even one fixed linear input pattern $x(t)=0.2t-2.5$, because the plant nonlinearity. Figure 9 illustrates the result from a recurrent neural network with feedback error $e(t)$ and $x(t)$ as inputs. The neural network is trained by backpropagation error through the identification network. Compared to the fixed gain PI controller, the neural network improves the tracking ability significantly.

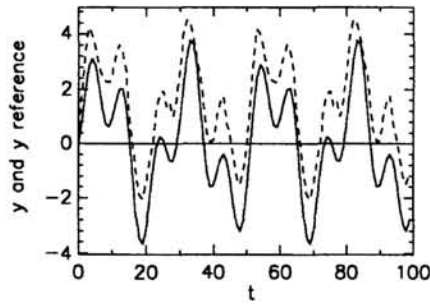

Figure 7 Tracking Result for Shifted Plant, plant output (dash line) and reference output (solid line).

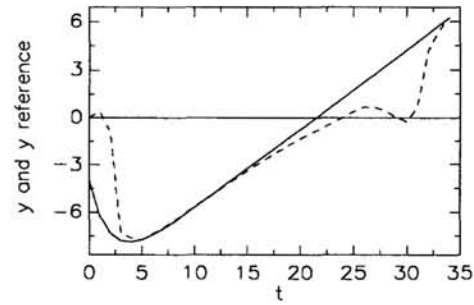

Figure 8 Reference Model Output (solid line) and PID Controlled Plant Output (dashed line)

The control surface of the updating output $f(.)$ is shown in Figure 10, which is the output from the neural network controller without recurrent link (see equation (3)). We plot the surface of the updating output from the controller with respect to input $x(t)$ and error feed back input $e(t)$. The gain of the controller is equivalent to the updating output from the network when error=1.0. As shown in the figure, the gain in the neighborhood about $x(t)=0$ changes largely according to the direction of changes in the plant in the corresponding region. The updating surface for a PID controller is a plane. The neural network implements a nonlinear continuous control gain surface.

For a more complicated case, we add $x(t-1)$ as another input to the neural network as well as $e(t-1)$, and train by error backpropagation through the identification network. These two inputs, $x(t)$ and $x(t-1)$ add difference information to the network. The network can adapt to not only different operating regions indicated by $x(t)$, but also different input patterns. Figure 11 shows the tracking results with two different input patterns. In Figure 11 (a), input pattern is $x(t)=4.0\sin(t/4.0)$. In Figure 11 (b) input pattern is $x(t)=\sin(2\pi t/25)+\sin(2\pi t/10)$.

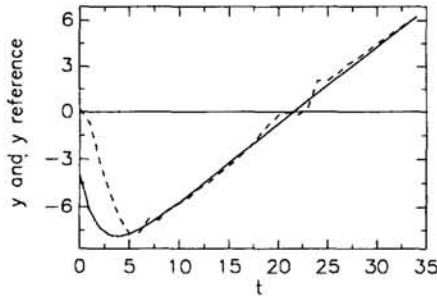
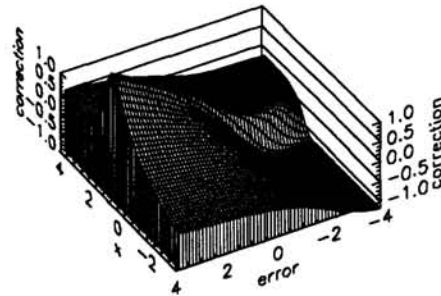

Figure 9 Reference Model Output (solid line) and
Neural Network Controled Output (dashed line)

Figure 10 Feedback Neural Controller Updating Surface

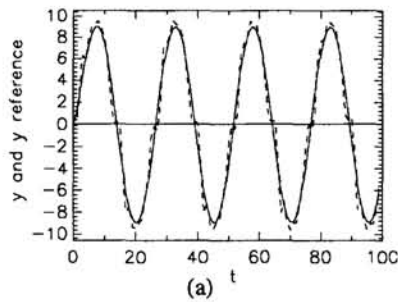
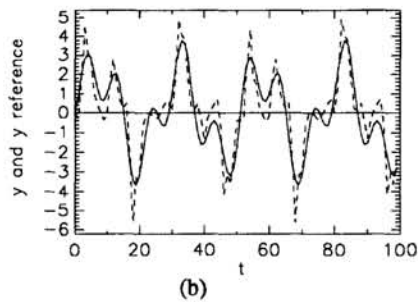

(a)    t              (b)

Figure 11 Output of the Reference Model (solid line) and the Plant (dash line)

## 3.3 INTEGRATED NEURAL CONTROLLER

As shown in the above section, when only error feedback neural controller is used, the control result is not very accurate. Now we combine feed forward and feedback control to realize good tracking and adaptation. Figure 12 shows the control result from the integrated controller when the plant is shifted 0.5. Compared to only feed forward control(Figure 7), the integrated controller has much better adaptation to the shifted plant.

When the plant changes, adding an extra feed back controller can avoid on-line training of feed forward network which may induce potential instability, and the adaptation is achieved. The output from the feedback network controller is driven by the error between the reference model and the plant.

## 4 DISCUSSIONS

We have emphasized in the above sections that a feed forward controller with only state feedback does not adapt when model uncertainties or noise/disturbance are present. The presence of a feed back controller can make the on line training of the feed forward network unnecessary, thus avoiding potential instability. The main reason for the instability of on-line training is the incompleteness of sample sets, which is referred to as a lack of persistent excitation in control theory[6]. First, it leads to an inaccurate identification network. Training with this network can result in an unstable controller. Second, it makes the training of controller away from global representation. With an error feedback adaptive network, the output from the feedback network controller is driven by the error between the reference model and the plant. In the simplest case when all the activity functions are linear and only the feedback errors are inputs, this kind of neural network is equivalent to a PID controller. However,

beyond the scope of PID controllers, the neural networks are capable to approximating nonlinear time variant control gain surfaces corresponding to different operating regions. Also, unlike a PID controller, the coefficients for the neural adaptive controller are obtained through a training procedure.

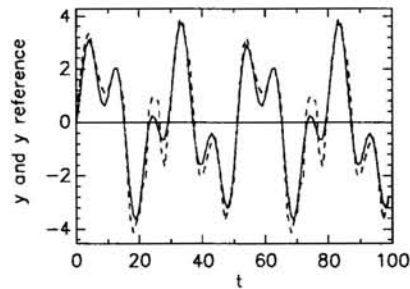

Figure 12 Integrated Network Controller Tracking Result for Shifted Plant,
Plant Output (dash line) and Reference Output (solid line).

The error feedback network behaves as a gain scheduling controller. It has rise time, overshoot consideration and delay problem. Feed forward control can compensate for these problems to some degree. For example, the feed forward network can perform a nonlinear mapping with designed time delay. Therefore with the feed forward network, the delay problem maybe overcame significantly. Also the feed forward controller can help to reduce rise time compare to use only feedback controller.

With the feed forward network, the feedback network controller can have much smaller gains compared to using a feedback network alone. This increases the noise rejection ability. Also this reduces the overshoot as well as settle time.

The neural network control architecture offers an alternative to the conventional approach. It gives a generic model for the broadest class of systems considered in control theory. However this model needs to be configured depending on the details of the control problem. With different inputs, the neural network controllers establish different internal hyperstates. When plant states are fed back to the network, a feed forward mapping is established as a function of the plant states by the neural network controller. When the errors between the reference model and the plant are used as the error feedback inputs to a dynamic neural network controller, the network functions as an associative memory nonlinear gain scheduled controller. The above two kinds of neural controller can be combined and complemented to achieve accurate tracking and adaptation.

## References

[1] Kumpati S. Narendra and Kannan Parthasarathy, "Gradient Methods for the Optimization of Dynamical Systems Containing Neural Networks," IEEE Trans. Neural Networks, vol. 2. pp252-262 Mar. 1991

[2] Psaltis, D., Sideris, A. and Yamamura, A., "Neural controllers," Proc. of 1st International Conference on Neural Networks, Vol. 4, pp551-558, San Diego, CA, 1987

[3] G. Lightbody, Q. H. Wu and G. W. Irwin, "Control applications for feed forward networks," Chapter 4, Neural Networks for Control and Systems, Edited by K.warwich, G. W. Irwin and K. J. Hunt 1992

[4] R. Abikowski and P. J. Gawthrop, "A survey of neural networks for control" Chapter 3, Neural Networks for Control and Systems, ISBN 0-86341-279-3, Edited by K.warwich, G. W. Irwin and K. J. Hunt 1992

[5] John Hertz, Anders Krogh and Richard G. Palmer, "Introduction to the Theory of Neural Computation,"

[6] Thomas Miller, Richard S. Sutton and Paul J. Werbos, "Neural Networks for Control"
